# Can Simple Cells Learn Curves? A Hebbian Model in a Structured Environment

**William R. Softky**
Divisions of Biology and Physics
103-33 Caltech
Pasadena, CA 91125
bill@aurel.caltech.edu

**Daniel M. Kammen**
Divisions of Biology and Engineering
216-76 Caltech
Pasadena, CA 91125
kammen@aurel.cns.caltech.edu

## ABSTRACT

In the mammalian visual cortex, orientation-selective 'simple cells' which detect straight lines may be adapted to detect curved lines instead. We test a biologically plausible, Hebbian, single-neuron model, which learns oriented receptive fields upon exposure to unstructured (noise) input and maintains orientation selectivity upon exposure to edges or bars of all orientations and positions. This model can also learn arc-shaped receptive fields upon exposure to an environment of only circular rings. Thus, new experiments which try to induce an abnormal (curved) receptive field may provide insight into the plasticity of simple cells. The model suggests that exposing cells to only a single spatial frequency may induce more striking spatial frequency and orientation dependent effects than heretofore observed.

## 1   Introduction

Although most mathematical theories of cortical function assume plasticity of individual cells, there is a strong debate in the biological community between "instructional" (plastic) and "selectional" (hard-wired) models of orientation-selective cells

(which we will call "simple cells") in striate visual cortex. Thus, a theory of simple cell learning which can make experimental predictions is desirable.

## 1.1    Overview of Plasticity Experiments

The most illuminating experiments addressing the plasticity of visual cortex are collectively called "stripe-rearing." Such experiments artificially restrict the visual environment of animals (usually kittens) to a few straight, dark, parallel lines (e.g. 3 vertical stripes.) In the many cases studied, examination of the visual cortex reveals that animals which viewed such limited visual environments posses more simple cells tuned to the exposed orientation than tuned to other orientations. (For comparison, the simple cells of animals with normal visual experience are equally distributed among all orientations.) But the observed changes in cell populations can be equally well explained by "instructional" and "selectional" hypotheses (Stryker *et al.*1978).

Although many variations on stripe-rearing have been tried (different orientations for each eye, one eye closed, etc.), only environments spanning a very restricted subset (straight lines) of the natural environment have been studied (Hirsch *et al.* 1983, Blakemore *et al.* 1978, and see references therein). Conclusions regarding plasticity have been based on changes in populations of simple cells, rather than on changes in individual cells. Statistical arguments based on changes in large groups of cells are questionable, since the well-documented lateral interactions between cortical neurons may constrain population ratios, e.g. limit the fraction of neurons responding to a single orientation.

## 1.2    New Experimental Approach

We propose several experiments to alter the receptive field (RF) of a single cell (see also Fregnac *et al.* 1988). How might that be done? The RF of a simple cell has only one characteristic spatial frequency (Jones & Palmer 1987 and ref's therein). To try altering the shape of that RF, it is necessary to present a pattern which is different from a simple bar or edge, but is still sufficiently similar in spatial frequency to activate the same population of retinal cells that detect the bar. An arc-shaped RF satisfies this condition; to generate an arc-shaped RF, an environment of circular rings (rather than bent bars) is necesary, since complete circles lack sharp end-effects which could overexcite spatial opponent cells and thus disturb learning.

This paper proposes a very simple Hebbian model of a neuron, and examines the resulting plasticity upon exposure to edge, bar, and arc-shaped stimuli.

## 2    Mathematical Model

The model applies a simple Hebbian learning rule to an array of about 400 synapses. There are several important features of this model. One is that the stimulus is a visual environment of structured input (bars, edges, or circles) rather than only stochastic (noise) input, as was used in the previous Hebb-learning models of Linsker (1986) and Kammen & Yuille (1988). (For a review of Hebbian learning and neural development see Kammen and Yuille 1990). Second, the input is Laplace filtered

to simulate the retinal processing stage; and third, all connections are rectified to be excitatory, like direct afferent input to simple cells.

## 2.1   Overview

We model the neuron as an array of non-negative synapses, distributed within a circular region. To let the neuron "see" a single pattern in the visual environment (see Figure 1, end of text), the array is overlaid on a much larger positive array (the filtered image), which represents the environment. Each synapse value is multiplied by its corresponding input pixel, and the sum of these products forms the neuron's "output." If the output is above a threshold value, each synapse is changed slightly to make it more like its corresponding pixel (the synapse is increased for a positive pixel, and decreased for a zero pixel.) If the output is low, nothing is changed. This process implements the correlation-based ("Hebbian") learning rule for synapse modification. To ensure maturation, we presented roughly one million training images to each neuron. Because there are many filtered images, only one is chosen at random for each iteration, and the neuron is overlapped at some random spatial offset.

## 2.2   Input Filtering Process

The visual environment is a collection of $N$ black-on-white pictures of a single shape (such as straight lines), at fixed contrast. The environment seen by the neuron is a set of $N$ filtered images, whose non-negative elements are produced from the pictures by a rectified, Laplace-like, center-surround process similar to that of the mammalian retina (Van Essen & Anderson 1988). To determine the RF of a mature array of synapses, the combined efficacy of all synapses is calculated for each pixel, and displayed as a grey scale (white = excitatory, black = inhibitory). See Figure 2, at end of text, for several examples of mature RF's.

## 2.3   Plasticity Under Visual Stimulation

The neuron's input synapses cover a circle much smaller than the filtered image. A single exposure to the environment overlaps the synapse array at a random position on the input image (chosen randomly from the training set). This overlap pairs each synapse with an input from a filter whose center has like polarity (on or off), so that each synapse represents a definite polarity of retinal cell.

A typical run involves perhaps $10^6$ exposures. There is no time variable, so that motion and temporal correlations between images are entirely absent. During each exposure a Hebb rule (section 2.4) changes synaptic weights based on current cell output and input values. When the neuron is exposed to filtered stochastic input ("noise-rearing"), synapses are intitialized randomly. When the neuron is exposed to structured environments, synapses are initialized with the orderly synapse arrays which *result* from noise-rearing. (As in animals, synapses may evolve in response to filtered random input before they are exposed to the external environment.)

## 2.4   A Choice of Hebb Rules for Learning Plasticity

Hebb postulated (1949) that neurons modify their synapses according to the following rule: the synapse will increase in efficacy if the post-synaptic and presynaptic excitations are coincident. There are many different formulae which satisfy Hebb's criterion; this model explores some simple representative ones. During each exposure to input, the synapses are adjusted according to the following type of hard-limited Hebb rule:

$$out \; = \; \sum_i syn_i \times in_i \tag{1}$$

And if $(out - thresh) > 0$ :

$$\Delta syn_i \;\; = \;\; (out - thresh)^n \times in_i \times growth \tag{2}$$
$$\text{if } in_i > 0 \text{ and } syn_i < 10$$
$$= \;\; -(out - thresh)^n \times decay \tag{3}$$
$$\text{if } in_i = 0 \text{ and } syn_i > 0.5$$
$$= \;\; 0 \;\; \text{otherwise} \tag{4}$$

The constants *growth* and *decay* are positive, and the exponent $n$ is at least one. Both types of threshold depend on the neuron's recent output history: either the average of the previous 200 outputs, or one half the maximum previous output (decaying by .9995 each exposure until a new $\frac{max}{2}$ exceeds it). This Hebb Rule assumes that the cell can detect the current input value *before* its modification by a synapse.

## 2.5   Choice of Parameters

The constants *growth* and *decay* are not sensitive parameters. We found that only three parameter regimes exist: all synapses saturate at maximum, all saturate at minimum, or some at maximum and some at minimum. Only the latter regime is of interest, because only it contains structured RF's.

Most simulations used $n = 1, 2, 3$ with both thresholds. The threshold based on maximum output enhances learning selectivity, while the averaged output version can be derived from a principle of "excess information" (See Appendix). Because simple cell RF's have approximately Gaussian envelopes (Jones & Palmer 1987), some simulations were done with Gaussian envelopes modulating the maximum synapse values. That modification made no difference in the results observed.

# 3   Results and Discussion

The production of oriented RFs during exposure to unstructured input confirms previous results by Linsker (1986) and Yuille *et al.* (1989), but with some important differences. Like those models, the neurons simulated here learn oriented stripe-patterns as a kind of lowest-energy configuration under exposure to spatially

correlated inputs. But unlike those models, we do not use: inhibitory connections or synapses; a synaptic-density gradient; a global conservation of synapse strength; or adjustable free parameters which can yield differently-shaped RFs. (In Linsker 1986 the ratio of "on" to "off" synapses is an adjustable parameter ; here, on and off pixels are represented equally.) Also, unlike previous models, mature RF's could have more than 3 lobes, depending on the ratio of filter size to RF size (Figure 2).

Under exposure to images of bars at all orientations, the neuron developed a mature RF matching a single one of them. Under exposure to stripes of nearly a single orientation, development of a mature RF depended on the stripes' spatial frequency.

In all cases, input patterns were learned much more quickly and strongly when their spatial frequency corresponded to the frequency of the Laplace filters. For input frequencies near the filter frequency, the resulting RF had a spatial frequency intermediate between the two. Otherwise, no learning occured unless the input frequency was a harmonic of the filter frequency, in which case the filter frequency was learned. Thus, this model predicts that enhanced learning might take place in kittens exposed to stripes of a single frequency, if that frequency is typical of simple-cell RF frequencies.

Under exposure to arcs or circles (with diameter $\approx 3 \times$ annular width), the model consistently developed RF's which matched a portion of the circle. These results suggest that animals which see only circles of a certain scale during the critical period may develop curved RFs (Barrow 1987) which differ qualitatively from those observed by such experiments as Jones & Palmer's (1987), who report seeing *no* curved contours in their point-by-point mappings of the RFs of normally-reared kittens. As with the stripes, the circles' annular width determines the spatial frequency of the retinal and simple cells which will respond best.

Such predictions must be treated with caution, because this paper does *not* simulate any version of the competing "selectional" model. It is possible that some of the effects predicted here for the "instructional" Hebbian model could also be observed by a "selectional" system.

To experimentally observe such effects in laboratory animals, many other known biological influences (eye acuity, interneuron effects, etc.) must be accounted for. We consider such problems elsewhere (Softky & Kammen *in preparation*), because they are of secondary importance to the striking and robust results of the model.

In summary, we have a single-cell model which contains essential biological features (such as all-excitatory input and synapses, and no global renormalizations). This model developes mature, oriented receptive fields under exposure to stochastic input for a wide variety of Hebb rules and for all non-trivial parameter regimes studied, with no apparent limitations on the number of lobes learned. Under exposure to structured input characteristic of normal environments, the model maintains oriented RF's; under exposure to input of "resonant" spatial frequency, the model develops RF's which reflect any novel orientation, spatial frequency, or curvature of the stimuli. This general, rule-independent response to the spatial frequency of

a stimulus – and the specific mechanism for generating abnormally curved RF's – may be useful in deciding experimentally whether simple cortical cells are indeed modifiable by Hebbian mechanisms.

This model does not attempt to explain curve-detection in a normal visual system. We already know that normal simple cells are not tuned for curves, and there are credible theories of normal curve-detection (Dobbins *et al.* 1987.) Rather, this model proposes using stimuli tuned to the natural spatial frequency of simple cells to induce a RF property which is distinctly abnormal, in order to better understand the rules by which normal visual properties emerge.

## 4   Appendix – Choice of Thresholds for the Hebb Rule

The choice of the average output as a threshold for a Hebb rule can be interpreted as follows. Consider a developing neuron whose output is the sum of $N$ inputs, each of which has independent probability distribution of mean $\alpha$ and standard deviation $\sigma$. We can calculate the information content in that sum, whose value has probability distribution (from the central limit theorem) of

$$P(out) \quad \propto \quad \exp\left(\frac{-(out - \langle out \rangle)^2}{2\sigma^2}\right). \tag{5}$$

The Shannon information (Shannon & Weaver 1962) carried by the sequence is

$$H(event) \quad = \quad -\ln P(event). \tag{6}$$

The excess information above the information carried by the average is thus

$$H_{excess} \quad = \quad H(out) - H(<out>) \tag{7}$$

$$\propto \quad \frac{(out - \langle out \rangle)^2}{2\sigma^2} \tag{8}$$

Thus, a Hebb rule using $n = 2$ and $thresh = \langle out \rangle$ is equivalent to learning based on the excess information carried in the output of an immature neuron.

The alternate threshold ($\frac{1}{2}max$) enhances selective learning for the following reason. If we consider the whole ensemble of patterns and shifts, the output characteristic which best distinguishes a matched synapse pattern from a random one is not its average output (the two averages are comparable for the all-excitatory case), but its maximum output. Thus, if a neuron can only 'remember' one characteristic number to serve as a threshold, then a number which changes during evolution (e.g. the maximum output) will refine selectivity more than one which is relatively constant. In addition, storing a maximum rather than an average removes the need to compute a running average, allowing unhindered evolution even after long periods of no input.

## Acknowledgements

D.K. is a Weizmann Postdoctoral Fellow and acknowledges support from the Weizmann Foundation, the James S. McDonnell Foundation and a NSF Presidential Young Investigator Award to Christof Koch.

# References

Barrow, H. (1987) "Learning Receptive Fields." *First I.E.E.E. Conference on Neural Networks*, **IV**, 115-121.

Blakemore C., Movshon J.A., & Van Sluyters R.C. (1978) "Modification of the Kitten's Visual Cortex by Exposure to Spatially Periodic Patterns." *Exp. Brain Res.*, **31**, 561-572.

Dobbins A., Zucker S. & Cynader M. (1987) "Endstopped Neurons in the Visual Cortex as a Substrate for Calculating Curvature." *Nature*, **329**, 438-441.

Fregnac Y., Shultz D., Thorpe S. & Bienenstock E. (1988) "A cellular analog of visual cortical plasticity." *Nature*, **333**, 367-370.

Hebb, D.O. (1949) "The Organization of Behavior: A Neuropsychological Theory." Wiley & Sons, New York.

Hirsch H., Leventhal A., McCall M. & Tieman D. (1983) "Effects of Exposure to Lines of One or Two Orientations on Different Cell Types in Striate Cortex of Cat." *J. Physiol.*, **337**, 241-255.

Jones J. & Palmer L. (1987) "The Two-Dimensional Spatial Structure of Simple Receptive Field in Cat Striate Cortex." *J. Neurophys.*, **58**, 1187-1232.

Kammen D.M. & Yuille A. (1988) "Spontaneous Symmetry-Breaking Energy Functions and the Emergence of Orientation Selective Cortical Cells." *Biol. Cybern.*, **59**, 23-31.

Kammen D.M. & Yuille A. (1990) "Self-Organizing Networks of Neural Units: Hebbian Learning in Development and Biological Computing." In:*Advances in Control Networks and Large Scale Distributed Processing Models*, Ablex Publishing, New Jersey.

Linsker R. (1986) "From basic network principles to neural architecture: Emergence of orientation-selective cells." *Proc. Natl. Acad. Sci. USA*, **83**, 8390-8394.

Shannon C. & Weaver W (1962) *The Mathematical Theory of Communication*, Univ. of Illinois Press, Urbana.

Stryker M., Sherk H., Leventhal A. & Hirsch H. (1978) "Physiological Consequences for the Cat's Visual Cortex of Effectively Restricting Early Visual Experience with Oriented Contours." *J. Neurophys.*, **41**, 896-909.

Van Essen D. & Anderson C. (1988) "Information Processing Strategies and Pathways in the Primate Retina and Visual Cortex." In: *Intro. to Neural and Electronic Networks*, Academic Press, Florida.

Yuille A., Kammen D.M. & Cohen D. (1989) "Quadrature and the Development of Orientation Selective Cortical Cells by Hebb Rules." *Biol. Cybern.*, **61**, 183-194.

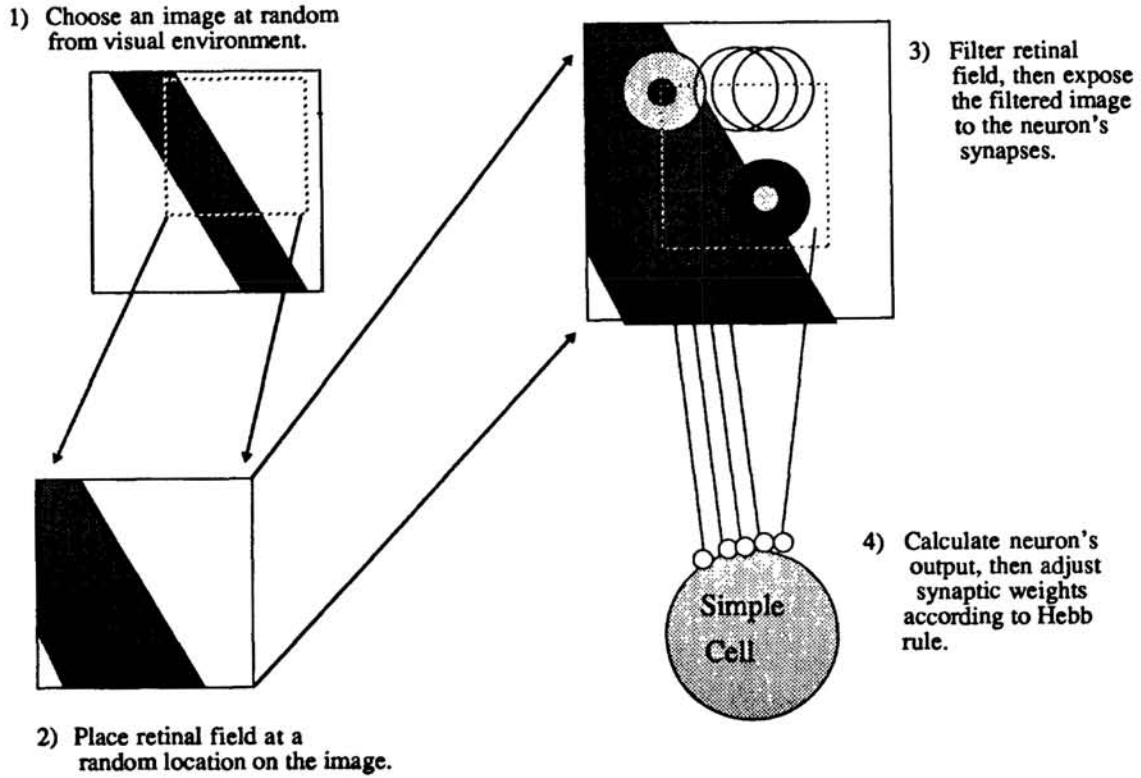

1) Choose an image at random from visual environment.

3) Filter retinal field, then expose the filtered image to the neuron's synapses.

4) Calculate neuron's output, then adjust synaptic weights according to Hebb rule.

Simple Cell

2) Place retinal field at a random location on the image.

**Figure 1:** Synapses Change Slightly During Each of a Million Iterations

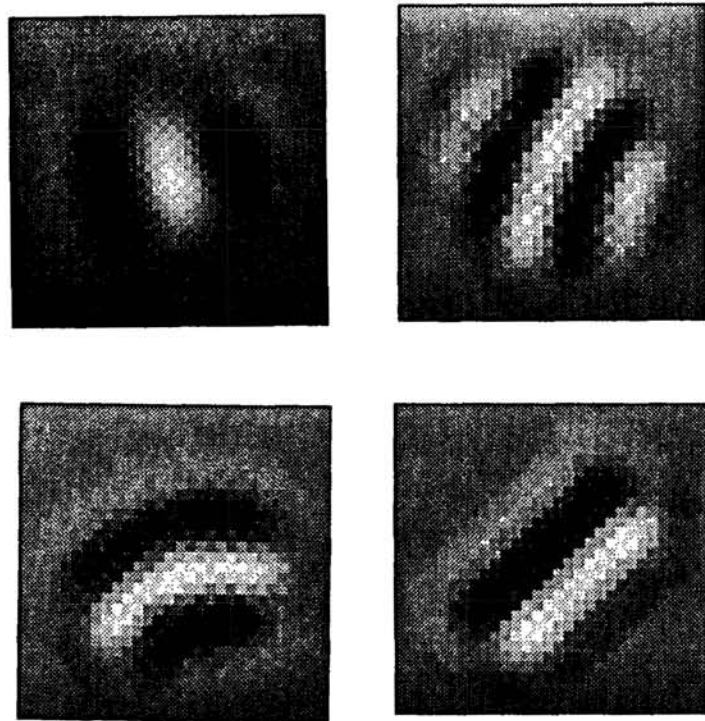

**Figure 2:** Learned Receptive Fields. Top row: Random pixel input, large (l) and small (r) filter sizes. Bottom row: Structured input, circular rings (l) and edges at different orientations (r).